# Stochastic Neurodynamics

**J.D. Cowan**
Department of Mathematics, Committee on
Neurobiology, and Brain Research Institute,
The University of Chicago, 5734 S. Univ. Ave.,
Chicago, Illinois 60637

## Abstract

The main point of this paper is that stochastic neural networks have a mathematical structure that corresponds quite closely with that of quantum field theory. Neural network Liouvillians and Lagrangians can be derived, just as can spin Hamiltonians and Lagrangians in QFT. It remains to show the efficacy of such a description.

## 1 INTRODUCTION

A basic problem in the analysis of large-scale neural network activity, is that one can never know the initial state of such activity, nor can one safely assume that synaptic weights are symmetric, or skew-symmetric. How can one proceed, therefore, to analyse such activity? One answer is to use a "Master Equation" (Van Kampen, 1981). In principle this can provide statistical information, moments and correlation functions of network activity by making use of ensemble averaging over all possible initial states. In what follows I give a short account of such an approach.

### 1.1 THE BASIC NEURAL MODEL

In this approach neurons are represented as simple gating elements which cycle through several internal states whenever the net voltage generated at their activated post-synaptic

sites exceeds a threshold. These states are "quiescent", "activated", and "refractory", labelled 'q', 'a', and 'r' respectively. There are then four transitions to consider: $q \rightarrow a$, $r \rightarrow a$, $a \rightarrow r$, and $r \rightarrow q$. Two of these, $q \rightarrow a$, and $r \rightarrow a$, are functions of the neural membrane current. I assume that on the time scale measured in units of $\tau_m$, the membrane time constant, the instantaneous transition rate $\lambda(q \rightarrow a)$ is a smooth function of the input current. $J_i(T)$. The transition rates $\lambda(q \rightarrow a)$ and $\lambda(r \rightarrow a)$ are then given by:

$$\lambda_q = \theta[(J(T)/J_q)-1] = \theta_q[J(T)], \qquad (1)$$

and

$$\lambda_r = \theta[(J(T)/J_r)-1] = \theta_r[J(T)], \qquad (2)$$

respectively, where $J_q$ and $J_r$ are the threshold currents related to $\theta_q$ and $\theta_r$, and where $\theta[x]$ is a suitable smoothly increasing function of x, and $T = t/\tau_m$. . The other two transition rates, $\lambda(a \rightarrow r)$ and $\lambda(r \rightarrow q)$ are defined simply as constants $\alpha$ and $\beta$. Figure 1 shows the "kinetic" scheme that results. Implicit in this scheme is the smoothing of input current pulses that takes place in the membrane,and also the smoothing caused by the

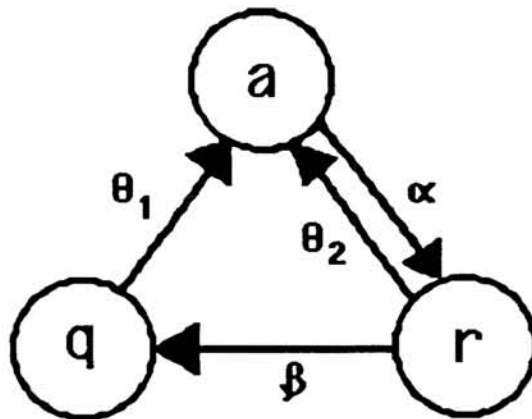

**Figure 1.** Neural state transition rates

presumed asynchronous activation of synapses. This simplified description of neural state transitions is essential to our investigation of cooperative effects in large nets.

## 1.2 PROBABILITY DISTRIBUTIONS FOR NEURAL NETWORK ACTIVITY

The configuration space of a neural network is the space of distinguishable patterns of neural activity. Since each neuron can be in the state q, a or r, there are $3^N$ such patterns in a network of N neurons. Since N is $0(10^{10})$, the configuration space is in principle very large. This observation, together with the existence of random fluctuations of neural

activity, and the impracticability of specifying the initial states of all the neurons in a large network, indicates the need for a probabilistic description of the formation and decay of patterns of neural activity.

Let $Q(T)$, $A(T)$, $R(T)$ denote the numbers of quiescent, activated, and refractory neurons in a network of N neurons at time T. Evidently,

$$Q(T)+A(T)+R(T) = N, \qquad (3)$$

Consider therefore N neurons in a d-dimensional lattice. Let a neural state vector be denoted by

$$| \Omega > = |v_1 v_2......... v_N> \qquad (4)$$

where $v_i$ means the neuron at the site i is in the state $v = q, a$ , or r. Let $P[\Omega(T)]$ be the probability of finding the network in state $| \Omega >$ at time T, and let

$$| P(T)\rangle = \sum_{\Omega} P[\Omega(T)]| \Omega> \qquad (5)$$

be a neural probability state vector.  Evidently $\sum_{\Omega} P[\Omega(T)] = 1.$   (6)

## 1.3 A NEURAL NETWORK MASTER EQUATION

Now consider the most probable state transitions which can occur in an asynchronous noisy network.  These are:

$(Q, A, R) \to (Q, A, R)$   no change
$(Q+1, A-1, R) \to (Q, A, R)$   activation of a quiescent cell
$(Q, A-1, R+1) \to (Q, A, R)$   activation of a refractory cell
$(Q, A+1, R-1) \to (Q, A, R)$   an activated cell becomes refractory
$(Q-1, A, R+1) \to (Q, A, R)$   a refractory cell beomes quiescent.

All other transitions,  e.g., those involving two or more transitions in time dT, are assumed to occur with probability O(dT).

These state transitions can be represented by the action on a set of basis vectors, of certain matrices. Let the basis vectors be:

$$|q>=\begin{pmatrix} \cdot \\ \cdot \\ 1 \end{pmatrix}, \qquad |a>=\begin{pmatrix} 1 \\ \cdot \\ \cdot \end{pmatrix}, \qquad |r>=\begin{pmatrix} \cdot \\ 1 \\ \cdot \end{pmatrix} \qquad (7)$$

and consider the Gell-Mann matrices representing the Lie Group SU(3) (Georgi, 1982) :

$$\lambda_1 = \begin{pmatrix} \cdot & 1 & \cdot \\ 1 & \cdot & \cdot \\ \cdot & \cdot & \cdot \end{pmatrix} \qquad \lambda_2 = \begin{pmatrix} \cdot & -i & \cdot \\ i & \cdot & \cdot \\ \cdot & \cdot & \cdot \end{pmatrix} \qquad \lambda_3 = \begin{pmatrix} 1 & \cdot & \cdot \\ \cdot & -1 & \cdot \\ \cdot & \cdot & \cdot \end{pmatrix} \qquad \lambda_4 = \begin{pmatrix} \cdot & \cdot & 1 \\ \cdot & \cdot & \cdot \\ 1 & \cdot & \cdot \end{pmatrix}$$

$$\lambda_5 = \begin{pmatrix} \cdot & \cdot & -i \\ \cdot & \cdot & \cdot \\ i & \cdot & \cdot \end{pmatrix} \qquad \lambda_6 = \begin{pmatrix} \cdot & \cdot & \cdot \\ \cdot & \cdot & 1 \\ \cdot & 1 & \cdot \end{pmatrix} \qquad \lambda_7 = \begin{pmatrix} \cdot & \cdot & \cdot \\ \cdot & \cdot & -i \\ \cdot & i & \cdot \end{pmatrix} \qquad \lambda_8 = \frac{1}{\sqrt{3}}\begin{pmatrix} 1 & \cdot & \cdot \\ \cdot & 1 & \cdot \\ \cdot & \cdot & -2 \end{pmatrix} \qquad (8)$$

and the raising and lowering operators:

$$\Lambda_{\pm 1} = \frac{1}{2}(\lambda_4 \pm i\lambda_5), \quad \Lambda_{\pm 2} = \frac{1}{2}(\lambda_1 \pm i\lambda_2), \quad \Lambda_{\pm 3} = \frac{1}{2}(\lambda_6 \pm i\lambda_7). \qquad (9)$$

It is easy to see that these operators act on the basis vectors $|v>$ as shown in figure 2.

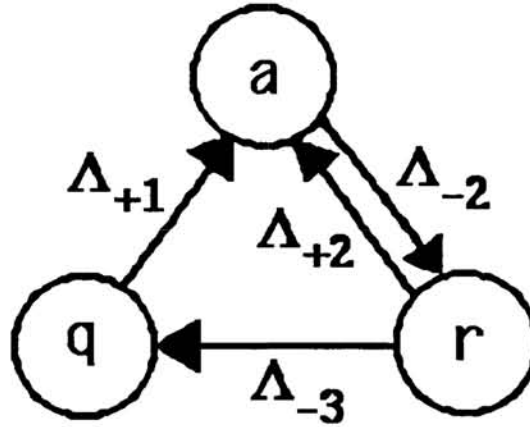

**Figure 2.** Neural State Transitions generated by the raising and lowering operators of the Lie Group SU(3).

It also follows that:
$$A = \sum_i \Lambda_{+1i}\Lambda_{-1i} = \sum_i \Lambda_{+2i}\Lambda_{-2i} \qquad (10)$$

and that:
$$J_i = \sum_j w_{ij}\Lambda_{+1j}\Lambda_{-1j} = \sum_j w_{ij}\Lambda_{+2j}\Lambda_{-2j}. \qquad (11)$$

The entire sequence of neural state transition into (Q,A,R) can be represented by the operator "Liouvillian":

$$L = \alpha \sum_i (\Lambda_{+2i} - 1) \Lambda_{-2i} + \beta \sum_i (\Lambda_{+3i} - 1) \Lambda_{-3i}$$

$$+ \frac{1}{N} \sum_i (\Lambda_{-1i} - 1) \Lambda_{+1i} \theta_q[J_i] + \frac{1}{N} \sum_i (\Lambda_{-2i} - 1) \Lambda_{+2i} \theta_r[J_i] . \qquad (12)$$

This operator acts on the state function $| P(T) \rangle$ according to the equation:

$$\frac{\partial}{\partial T} | P(T) \rangle = - L | P(T) \rangle. \qquad (13)$$

This is the neural network analogue of the Schrödinger equation, except that $P[\Omega(T)] = \langle \Omega | P(T) \rangle$ is a real probability distribution, and L is not Hermitian. In fact this equation is a Markovian representation of neural network activity (Doi, 1976; Grassberger & Scheunert, 1980), and is the required master equation.

## 1.4 A SPECIAL CASE: TWO-STATE NEURONS

It is helpful to consider the simpler case of two state neurons first, since the group algebra is much simpler. I therefore neglect the refractory state, and use the two dimensional basis vectors:

$$| q \rangle = \begin{pmatrix} . \\ 1 \end{pmatrix} , \qquad | a \rangle = \begin{pmatrix} 1 \\ . \end{pmatrix} \qquad (14)$$

corresponding to the kinetic scheme shown in figure 3a:

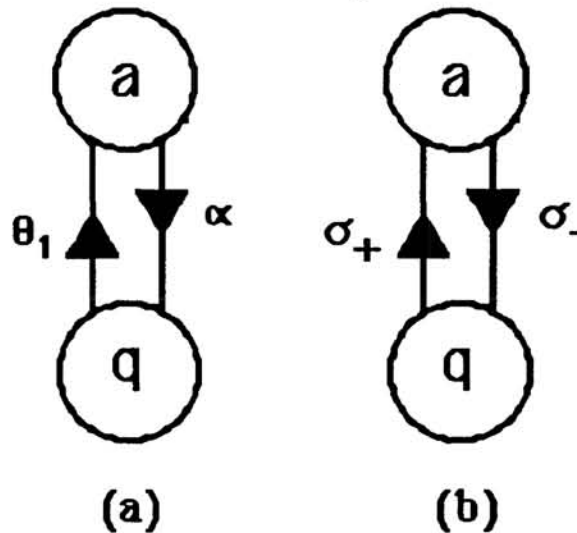

**(a)**                    **(b)**

**Figure 3.** (a) Neural State Transitions in the two-state case, (b) Neural State Transitions generated by the raising and lowering operators of the Lie Group SU(2).

The relevant matrices are the well-known Pauli spin matrices representing the Lie Group SU(2) (Georgi, 1982):

$$\sigma_1 = \begin{pmatrix} . & 1 \\ 1 & . \end{pmatrix} \qquad \sigma_2 = \begin{pmatrix} . & -i \\ i & . \end{pmatrix} \qquad \sigma_3 = \begin{pmatrix} 1 & . \\ . & -1 \end{pmatrix} \qquad (15)$$

and the raising and lowering operators:

$$\sigma_\pm = \frac{1}{2} \, (\sigma_1 \pm i\sigma_2) \qquad (16)$$

giving the state transiiton diagram shown in figure 3(b). The corresponding neural Liouvillian is:

$$L = \alpha \sum_i (\sigma_{+i} - 1) \, \sigma_{-i} \; + \; \frac{1}{N} \sum_i (\sigma_{-1} - 1) \, \sigma_{+i} \, \theta_q[J_i] \qquad (17)$$

where
$$J_i = \sum_j w_{ij} \, \sigma_{+j} \, \sigma_{-j} . \qquad (18)$$

Physicists will recognize this Liouvillian as a generalization of the Regge spin Hamiltonian of QFT:

$$L = \alpha \sum_i (\sigma_{+i} - 1) \, \sigma_{-i} \; + \; \frac{\kappa}{N} \sum_i \sum_j (\sigma_{-1} - 1) \, \sigma_{+i} \, \sigma_{+j} \, \sigma_{-j} . \qquad (19)$$

In principle, eqn. (13) with L given by eqn. (12) or (17), together with initial conditions, contains a complete description of neural network activity, since its formal solution takes the form:

$$| P(T)> = \exp \left\{ - \int_0^T L(T') dT' \right\} | P(0)> . \qquad (20)$$

## 1.5 MOMENT GENERATING EQUATIONS AND SPIN-COHERENT STATES

Solving this system of equation in detail however, is a difficult problem. In practice one is satisfied with the first few statistical moments. These can be obtained as follows (I describe here the two-state case. Similar but more complicated calculations obtain for the three-state case).

Consider the following "spin-coherent states" (Perelomov 1986; Hecht 1987):

$$< \alpha | = <0| \exp \left( \sum_i \alpha_i \sigma_{-i} \right), \qquad | \alpha > = \exp \left( \sum_i \alpha_i^* \sigma_{+i} \right) | 0 > \qquad (21)$$

where $\alpha$ is a complex number, and $< 0 |$ is the "vacuum" state $< q_1 q_2 \ldots\ldots q_N |$. Evidently

$$< \alpha \mid P > \; = \; < \alpha \mid \sum_{\Omega} P[\Omega(T)] \mid \Omega > \; = \; \sum_{\Omega} P[\Omega(T)] < \alpha \mid \Omega > .$$

It can be shown that $< \alpha \mid \Omega > \; = \; \alpha_1^{v_1} \, \alpha_2^{v_2} \, ............ \alpha_N^{v_N}$ and that $< \alpha \mid P > \; =$

$G(\alpha_1 \, \alpha_2 \, .... \, \alpha_N)$ the moment generating function for the probability distribution $P(T)$.

It can then be shown that:

$$\frac{\partial G}{\partial T} = [ \, \alpha \sum_i (D\alpha_i - 1) \frac{\partial}{\partial \alpha_i} \; + \; \frac{1}{N} \sum_i (\frac{\partial}{\partial \alpha_i} - 1) \, D\alpha_i \, \theta_q[J_i]] \, G \qquad (22)$$

where $\qquad D\alpha_i = \alpha_i (1 - \alpha_i \frac{\partial}{\partial \alpha_i})$ and $J_i = \sum_j w_{ij} \, D\alpha_j \frac{\partial}{\partial \alpha_j} .$ $\qquad$ (23)

i.e.; the moment generating equation expressed in the "oscillator-algebra" representation.

## 1.5 A NEURAL NETWORK PATH INTEGRAL

The content of eqns. (22) and (23) can be summarized in a Wiener-Feynman Path Integral (Schulman 1981). It can be shown that the transition probability of reaching a state $\Omega'$ $(T)$ given the initial state $\Omega(T_0)$, the so-called propagator $\mathbf{G}(\Omega', T \mid \Omega, T_0)$, can be expressed as the Path Integral:

$$\int \prod_i \mathbf{D}\alpha_i (T') \exp \left[ \; \int_T^{T_0} \{ \sum_i \frac{1}{2} \, (D'\alpha_i \, D\alpha_i^* - D\alpha_i \, D'\alpha_i^*) - L(D\alpha_i, D\alpha_i^*) \} \right], \qquad (24)$$

where $D'\alpha_i = \frac{\partial}{\partial T} D\alpha_i$ and $\mathbf{D}\alpha_i (T') = (\frac{2}{\pi})^n \lim_{n \to \infty} \prod_{j=1}^{n} \frac{d^2 \alpha_i (j)}{(1 + \alpha_i (j) \alpha_i^* (j))^3}$, and

where $d^2 \alpha = d(Rl \, \alpha) \, d(Im \, \alpha)$. This propagator is sometimes written as an expectation

with respect to the Wiener measure $\int \prod_i \mathbf{D}\alpha_i (T')$ as:

$$\mathbf{G}(\Omega' \mid \Omega) = < \exp \left[ - \int_T^{T_0} dT' \, \mathbf{L} \right] > \qquad (25)$$

where the neural network Lagrangian is defined as:

$$\mathcal{L} = L(D\alpha_i, D\alpha_i^*) - \sum_i \frac{1}{2} (D'\alpha_i D\alpha_i^* - D\alpha_i D'\alpha_i^*). \quad (26)$$

The propagator $\mathcal{G}$ contains all the statistics of the network activity. Steepest descent methods, asymptotics, and Liapunov-Schmidt bifurcation methods may be used to evaluate it.

# 2 CONCLUSIONS

The main point of this paper is that stochastic neural networks have a mathematical structure that corresponds quite closely with that of quantum field theory. Neural network Liouvillians and Lagrangians can be derived, just as can spin Hamiltonians and Lagrangians in QFT. It remains to show the efficacy of such a description.

**Acknowledgements**

The early stages of this work were carried out in part with Alan Lapedes and David Sharp of the Los Alamos National Laboratory. We thank the Santa Fé Institute for hospitality and facilities during this work, which was supported in part by grant # N00014-89-J-1099 from the US Department of the Navy, Office of Naval Research.

**References**

Van Kampen, N. (1981), Stochastic Processes in Physics & Chemistry (N. Holland, Amsterdam).
Georgi, H. (1982), Lie Algebras in Particle Physics (Benjamin Books, Menlo Park)
Doi, M. (1976), J.Phys. A. Math. Gen. **9**, 9, 1465-1477; 1479-1495
Grassberger, P. & Scheunert, M. (1980), Fortschritte der Physik **28**, 547-578.
Hecht, K.T. (1987), The Vector Coherent State Method (Springer, New York)
Perelomov, A. (1986), Generalized Coherent States and Their Applications (Springer, New York).
Matsubara, T & Matsuda, H. (1956). A lattice model of Liquid Helium, I. Prog. Theoret. Phys. **16**, 6, 569-582.
Schulman, L. (1981), Techniques and Applications of Path Integration (Wiley, New York).